# Graphical Gaussian Vector for Image Categorization

**Tatsuya Harada**
The University of Tokyo/JST PRESTO
7-3-1 Hongo Bunkyo-ku, Tokyo Japan
harada@isi.imi.i.u-tokyo.ac.jp

**Yasuo Kuniyoshi**
The University of Tokyo
7-3-1 Hongo Bunkyo-ku, Tokyo Japan
kuniyosh@isi.imi.i.u-tokyo.ac.jp

## Abstract

This paper proposes a novel image representation called a Graphical Gaussian Vector (GGV), which is a counterpart of the codebook and local feature matching approaches. We model the distribution of local features as a Gaussian Markov Random Field (GMRF) which can efficiently represent the spatial relationship among local features. Using concepts of information geometry, proper parameters and a metric from the GMRF can be obtained. Then we define a new image feature by embedding the proper metric into the parameters, which can be directly applied to scalable linear classifiers. We show that the GGV obtains better performance over the state-of-the-art methods in the standard object recognition datasets and comparable performance in the scene dataset.

## 1 Introduction

The Bag of Words (BoW) [7] is the de facto standard image feature for the image categorization. In a BoW, each local feature is assigned to the nearest codeword and an image is represented by a histogram of the quantized features. Several approaches inspired by a BoW have been proposed in recent years [9], [23], [28], [27], [29]. While it is well established that using a large number of codewords improves classification performance, the drawback is that assigning local features to the nearest codeword is computationally expensive. To overcome this problem, some studies have proposed building an efficient image representation with a smaller number of codewords [22], [24]. Finding an explicit correspondence between local features is another way of categorizing images using a BoW [4], [12], [26], and this approach has been improved by representing a spatial layout of local features as a graph [11], [2], [16], [8]. Explicit correspondences between features have an advantage over a BoW as information loss in the vector quantization can be avoided. However, the drawback with this approach is that the identification of corresponding points with minimum distortion is also computationally expensive. Therefore, the aim of our research is to build an efficient image representation without using codewords or explicit correspondences between local features, while still achieving high classification accuracy.

Since having a spatial layout of local features is important for an image to have semantic meaning, it is natural that embedding spatial information into an image feature improves classification performance [18], [5], [14], [17]. Several approaches take advantage of this fact, ranging from local (e.g., SIFT) to global (e.g., Spatial Pyramid). Meanwhile, we will focus on the spatial layout of local features, which is the midlevel of the spatial information.

In this paper, we model an image as a graph representing the spatial layout of local features and define a new image feature based on this graph, where a proper metric is embedded into the feature. We show that the new feature provides high classification accuracy, even with a linear classifier. Specifically, we model an image as a Gaussian Markov Random Field (GMRF) whose nodes correspond to local features and consider the GMRF parameters as the image feature. Although the GMRF is commonly used for image segmentation, it is rarely used in modern image categorization pipelines despite being an effective way of modeling the spatial layout. In order to extract the repre-

sentative feature vector from the GMRF, the choice of coordinates for the parameters and the metric between them needs to be carefully made. We define the proper coordinates and the metric from an information geometry standpoint [1] and derive an optimal feature vector. The resultant feature vector is called a Graphical Gaussian Vector.

The contributions of this study are summarized as follows: 1) A novel and efficient image feature is developed by utilizing the GMRF as a tool for object categorization. 2) This approach is implemented by developing the Graphical Gaussian Vector feature, which is based on the GMRF and the information geometry. 3) Using standard image categorization benchmarks, we demonstrate that the proposed feature has better performance over the state-of-the-art methods, even though it is not based on mainstream modules (such as codebooks and correspondence between local features). To the best of our knowledge, this is the first image feature for the object categorization that utilizes the expectation parameters of the GMRF with its Fisher information metric, and achieves a level of accuracy comparable to that of the codebook and local feature matching approaches.

## 2 Graphical Gaussian Vector

### 2.1 Overview of Proposed Method

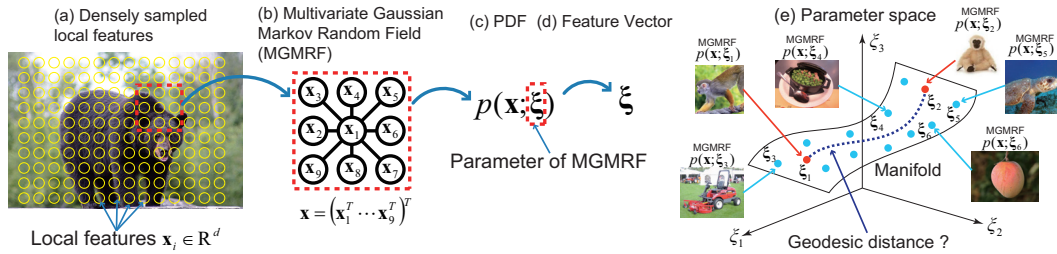

Figure 1: Overview of image feature extraction based on a multivariate GMRF.

In this section, we present an overview of our method. Initially, local features $\{\boldsymbol{x}_i \in \mathbb{R}^d\}_{i=1}^M$ are extracted using a dense sampling strategy (Fig. 1(a)). We then use a multivariate GMRF to model the spatial relationships among local features (Fig. 1(b)). The GMRF is represented as a graph $\mathcal{G}(\mathcal{V},\mathcal{E})$, whose vertices $\mathcal{V}$ and edges $\mathcal{E}$ correspond to local features and the dependent relationships between those features, respectively. Let the vector $\boldsymbol{x}$ be a concatenation of local features in $\mathcal{V}$ and let $\boldsymbol{\xi}_j$ be a parameter of the GMRF of an image $I_j$, the image $I_j$ can be represented by a probability distribution $p(\boldsymbol{x};\boldsymbol{\xi}_j)$ of the GMRF (Fig. 1(c)). We consider the parameter $\boldsymbol{\xi}_j$ of the GMRF to be a feature vector of the image $I_j$ (Fig. 1(d)). Assuming that $\boldsymbol{\xi}$ is a coordinate system, the whole probability distribution model can be considered as a manifold, where each probability distribution is represented as a point in that space (Fig. 1(e)). However, because the space spanned by parameters of a probability distribution is not a Euclidean space, we have to be very careful when choosing parameters for the probability distribution and the metric among them. We make use of concepts from the information geometry [1] and extract proper parameters and a metric from the GMRF. Finally, we define the new image feature by embedding the metric into the extracted parameters to build an image categorization system with a scalable linear classifier. In the following sections, we describe this process in more detail.

### 2.2 Image Model and Parameters

Given $M$ local features $\{\boldsymbol{x}_i \in \mathbb{R}^d\}_{i=1}^M$, the aim is to model a probability distribution of the local features representing the spatial layout of the image using the multivariate GMRF $\mathcal{G} = (\mathcal{V},\mathcal{E})$. First, a vector $\boldsymbol{x}$ is built by concatenating the local features corresponding to the vertices $\mathcal{V}$ of the GMRF. Let $\{\boldsymbol{x}_i\}_{i=1}^n$ are local features that we are focusing on, we obtain the concatenated vector as $\boldsymbol{x} = (\boldsymbol{x}_1^\top \cdots \boldsymbol{x}_n^\top)^\top$ (e.g., Fig. 1(b), where $n = 9$). Note that the dimensionality of $\boldsymbol{x}$ is $nd$ and does not depend on the number of local features $M$, the image size, or the aspect ratio. However, since all results valid for a scalar local feature are also valid for a multivariate local feature, in this section we consider the dimensionality of local features is 1 ($d = 1$) for simplicity. That is $\dim(\boldsymbol{x}) = n$.

Let $\boldsymbol{\mu} = \mathbb{E}[\boldsymbol{x}]$, $P = \mathbb{E}[(\boldsymbol{x} - \boldsymbol{\mu})(\boldsymbol{x} - \boldsymbol{\mu})^{\top}]$, and $J = P^{-1}$. A random vector $\boldsymbol{x}$ is called a Gaussian Markov Random Field (GMRF) with respect to $\mathcal{G} = (\mathcal{V}, \mathcal{E})$, if and only if its density has the form $p(\boldsymbol{x}) = (2\pi)^{-n/2}|J|^{1/2}\exp(-\frac{1}{2}(\boldsymbol{x} - \boldsymbol{\mu})^{\top}J(\boldsymbol{x} - \boldsymbol{\mu}))$ and $J_{ij} = 0$ for all $\{i, j\} \notin \mathcal{E}$. Because the Gaussian distribution can be represented as an exponential family, here, we consider an exponential family as follows:

$$p(\boldsymbol{x}) = \exp\left(\boldsymbol{\theta}^{\top}\boldsymbol{\phi}(\boldsymbol{x}) - \Phi(\boldsymbol{\theta})\right), \tag{1}$$

where $\boldsymbol{\theta}$ are the natural parameters, $\boldsymbol{\phi}(\boldsymbol{x})$ is the sufficient statistic, and $\Phi(\boldsymbol{\theta}) = \log \int \exp(\boldsymbol{\theta}^{\top}\boldsymbol{\phi}(\boldsymbol{x}))d\boldsymbol{x}$ is the log-normalizer. $\boldsymbol{\theta}$ and $\boldsymbol{\phi}(\boldsymbol{x})$ of the GMRF are obtained as [15]:

$$\theta^i = h_i,\ \theta^{ii} = -\frac{1}{2}J_{ii},\ \theta^{jk} = -J_{jk},\ (i \in \mathcal{V}, \{j, k\} \in \mathcal{E}), \tag{2}$$

$$\phi_i(\boldsymbol{x}) = x_i,\ \phi_{ii}(\boldsymbol{x}) = x_i^2,\ \phi_{jk}(\boldsymbol{x}) = x_j x_k,\ (i \in \mathcal{V}, \{j, k\} \in \mathcal{E}), \tag{3}$$

where $\boldsymbol{h} = J\boldsymbol{\mu}$. The expectation parameter $\boldsymbol{\eta} = \mathbb{E}[\boldsymbol{\phi}(\boldsymbol{x})]$ is an implicit parameterization belonging to the exponential family. The expectation parameters are obtained as [15]:

$$\eta_i = \mu_i,\ \eta_{ii} = P_{ii} + \mu_i^2,\ \eta_{jk} = P_{jk} + \mu_j \mu_k,\ (i \in \mathcal{V}, \{j, k\} \in \mathcal{E}). \tag{4}$$

The natural and expectation parameters can be transformed into each other [1]. They are called mutually dual as each is the dual coordinate system of the other. The two coordinate systems are closely related through the Fisher information matrices (FIMs) $G_{ij}(\boldsymbol{\theta})$ and $G^{*ij}(\boldsymbol{\eta})$: $G_{ij}(\boldsymbol{\theta}) = \frac{\partial \eta_i}{\partial \theta^j}$, and $G^{*ij}(\boldsymbol{\eta}) = \frac{\partial \theta^i}{\partial \eta_j}$, where $G^*(\boldsymbol{\eta}) = G^{-1}(\boldsymbol{\theta})$. If we take the natural parameters or the expectation parameters as a coordinate system for an exponential family, a flat structure can be realized [1]. In particular, $\boldsymbol{\theta}$ is called a 1-affine coordinate system, and the space spanned by $\boldsymbol{\theta}$ is called 1-flat. Similarly, $\boldsymbol{\eta}$ is called a (-1)-affine coordinate system, and the space spanned by $\boldsymbol{\eta}$ is called (-1)-flat. Those spaces are similar to a Euclidean space, but we need to be careful that the spaces spanned by the natural or expectation parameters are different from a Euclidean space, as the metrics vary for different parameters. We will discuss how to determine the metrics in those spaces in Sections. 2.4 and 2.5.

To summarize this section, the natural and expectation parameters are similar and interchangeable through the FIMs. By using these parameters, we can obtain flatness similar to the Euclidean space. Although it does not matter whether we choose natural or expectation parameters, we use expectation parameters (Eq. (4)) as feature vectors because they can be calculated directly from the mean and covariance of local features. We will see a multivariate extension of the GMRF and its calculation in the next section.

## 2.3 Calculation of Expectation Parameters

In this section, we describe the calculations of the expectation parameters of the multivariate GMRF. First, we define the graph structure of the GMRF. We use star graphs shown in Fig. 2, where four neighbors (Fig. 2(a)) or eight neighbors (Fig. 2(b)) are usually used. While a graph having more neighbors is obviously able to represent richer spatial information, the compact structure is preferable for efficiency. Therefore, we employed the approximated graph structures shown in Fig. 2(c), which represents the vertical and horizontal relationships among local features, and Fig. 2(d), which represents vertical, horizontal and, diagonal relationships.

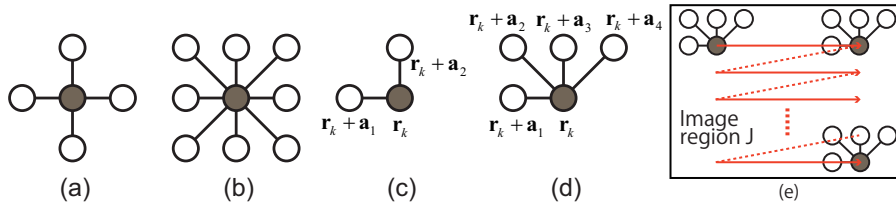

Figure 2: Structures of the GMRF.

Next, we show a method for estimating the expectation parameters of each image. In practice, Eq. (4) in a multivariate case can be determined by calculating the local auto-correlations of local

features. Here we present the detailed calculations of Eq. (4) using Fig. 2(c) as an example. Let $\boldsymbol{x}(\boldsymbol{r}_k) \in \mathbb{R}^d$ be the local feature at a reference point $\boldsymbol{r}_k$ and let $\boldsymbol{a}_i$ and $\boldsymbol{a}_j$ be the displacement vectors, which are defined by the structure of the GMRF. Then, the local auto-correlation matrices are obtained as: $C_{i,j} = \frac{1}{N_J} \sum_{k \in J} \boldsymbol{x}(\boldsymbol{r}_k + \boldsymbol{a}_i)\boldsymbol{x}(\boldsymbol{r}_k + \boldsymbol{a}_j)^\top$, where $N_J$ is the number of local features in the image region $J$. Especially if we define $\boldsymbol{a}_0 = \boldsymbol{0}$, $C_{0,i} = \frac{1}{N_J} \sum_{k \in J} \boldsymbol{x}(\boldsymbol{r}_k)\boldsymbol{x}(\boldsymbol{r}_k + \boldsymbol{a}_i)^\top$. Let a vector concatenating local features in the vertices at the reference point $\boldsymbol{r}_k$ be $\boldsymbol{x}_k^\top = (\boldsymbol{x}(\boldsymbol{r}_k)^\top \boldsymbol{x}(\boldsymbol{r}_k + \boldsymbol{a}_1)^\top \boldsymbol{x}(\boldsymbol{r}_k + \boldsymbol{a}_2)^\top)$, $P + \boldsymbol{\mu}\boldsymbol{\mu}^\top$ is calculated to be:

$$P + \boldsymbol{\mu}\boldsymbol{\mu}^\top = \frac{1}{N_J} \sum_{k \in J} \boldsymbol{x}_k \boldsymbol{x}_k^\top = \begin{pmatrix} C_{0,0} & C_{0,1} & C_{0,2} \\ C_{1,0} & C_{1,1} & C_{1,2} \\ C_{2,0} & C_{2,1} & C_{2,2} \end{pmatrix}. \tag{5}$$

The expectation parameters of the GMRF depicted in Fig. 2(c) can be obtained as: $\boldsymbol{\eta} = (\boldsymbol{\mu}_0^\top \, \boldsymbol{\mu}_1^\top \, \boldsymbol{\mu}_2^\top \, f^\top(C_{0,0}) \, f^\top(C_{1,1}) \, f^\top(C_{2,2}) \, g^\top(C_{0,1}) \, g^\top(C_{0,2}))^\top$, where $f(\cdot)$ returns a column vector consisting of the elements of the upper triangular portion of the input matrix, $g(\cdot)$ returns a column vector containing all the elements of the input matrix and $\boldsymbol{\mu}_i = \frac{1}{N_J} \sum_{k \in J} \boldsymbol{x}(\boldsymbol{r}_k + \boldsymbol{a}_i)$. Note that $C_{1,2}$ is omitted, because there is no edge between the vertices at $\boldsymbol{r}_k + \boldsymbol{a}_1$ and $\boldsymbol{r}_k + \boldsymbol{a}_2$. In general, the expectation parameters (Eq. (4)) on the star graph can be calculated by:

$$\boldsymbol{\eta} = \left( \boldsymbol{\mu}_0^\top \cdots \boldsymbol{\mu}_{n-1}^\top \, f^\top(C_{0,0}) \cdots f^\top(C_{n-1,n-1}) \, g^\top(C_{0,1}) \cdots g^\top(C_{0,n-1}) \right)^\top, \tag{6}$$

where $n = |\mathcal{V}|$ is the number of vertices. The dimensionality of $\boldsymbol{\eta}$ is: $nd + n(d+1)d/2 + (n-1)d^2$, where $d$ is the dimensionality of the local feature. Also note that $\{C_{i,j}\}_{i \neq j \wedge i,j \neq 0}$ can be omitted.

By scanning the image region $J$ (Fig. 2(e)), if we have enough local features, the means $\{\boldsymbol{\mu}_i\}_{i=0}^{n-1}$ and covariance matrices $\{C_{i,i}\}_{i=0}^{n-1}$ of local features in the region $J$ come to the vector $\boldsymbol{\mu}_0$ and matrix $C_{0,0}$, respectively. The expectation parameters (Eq. (4)) can be approximated by:

$$\boldsymbol{\eta} = (\underbrace{\boldsymbol{\mu}_0^\top \cdots \boldsymbol{\mu}_0^\top}_{n} \underbrace{f^\top(C_{0,0}) \cdots f^\top(C_{0,0})}_{n} \underbrace{g^\top(C_{0,1}) \cdots g^\top(C_{0,n-1})}_{n-1})^\top. \tag{7}$$

Equation (7) is calcuated more efficiently than Eq. (6) and comes to the same vector as Eq. (6). However, in the preliminary experiment, Eq. (6) is better than Eq. (7) in terms of the classification accuracies. In the following sections, we use Eq. (6) for the expectation parameters.

## 2.4 Metric

In Section 2.2, we mentioned that the metric varies depending on the parameters. We now derive a metric between the expectation parameters [1]. Let $ds$ represent the length of the small line-element connecting $\boldsymbol{\eta}$ and $\boldsymbol{\eta} + d\boldsymbol{\eta}$. $d\boldsymbol{\eta}$ is represented by using basis vectors $\boldsymbol{e}^{*i}$: $d\boldsymbol{\eta} = \sum_i \eta_i \boldsymbol{e}^{*i}$. The squared distance can be calculated by: $ds^2 = \langle d\boldsymbol{\eta}, d\boldsymbol{\eta} \rangle = \sum_{i,j} \langle \boldsymbol{e}^{*i}, \boldsymbol{e}^{*j} \rangle d\eta_i d\eta_j$, where $\langle \cdot, \cdot \rangle$ is the inner product of two vectors. By applying the Taylor expansion to KL divergence between $p(\boldsymbol{x}; \boldsymbol{\eta})$ and $p(\boldsymbol{x}; \boldsymbol{\eta} + d\boldsymbol{\eta})$, $ds^2$ can be represented as follows: $ds^2 = \mathrm{KL}[p(\boldsymbol{x}; \boldsymbol{\eta}) : p(\boldsymbol{x}; \boldsymbol{\eta} + d\boldsymbol{\eta})] = \frac{1}{2} d\boldsymbol{\eta}^\top G^* d\boldsymbol{\eta} = \frac{1}{2} \sum_{i,j} G^{*ij} d\eta_i d\eta_j$, where $G^*$ is the FIM. By comparing these equations, it is clear that the metric matrix consisting of the inner products of the basis vectors corresponds to the FIM:

$$G^{*ij} = \langle \boldsymbol{e}^{*i}, \boldsymbol{e}^{*j} \rangle. \tag{8}$$

Thus, the FIM is a proper metric for the feature vectors (the expectation parameters) obtained from the GMRF.

The Cramér-Rao inequality gives us a better understanding of the FIM. Assuming that $\hat{\boldsymbol{\eta}}$ is an unbiased estimator, the variance-covariance matrix of $\hat{\boldsymbol{\eta}}$ satisfies: $\mathrm{Var}[\hat{\boldsymbol{\eta}}] \geq \frac{1}{N}(G^*)^{-1}$. Consequently, the FIM is considered to be the inverse of the variance of an estimator, making it natural to use the matrix as a distance metric between the parameters.

## 2.5 Implementation of Graphical Gaussian Vector

At first, we build the concatenated vector as $\boldsymbol{x} = (\boldsymbol{x}_1^\top \cdots \boldsymbol{x}_n^\top)^\top$, where each $\boldsymbol{x}_i$ corresponds to the local feature of the vertex $i$. By using all training data, the mean $\boldsymbol{\mu} = \mathbb{E}[\boldsymbol{x}]$ and the precision

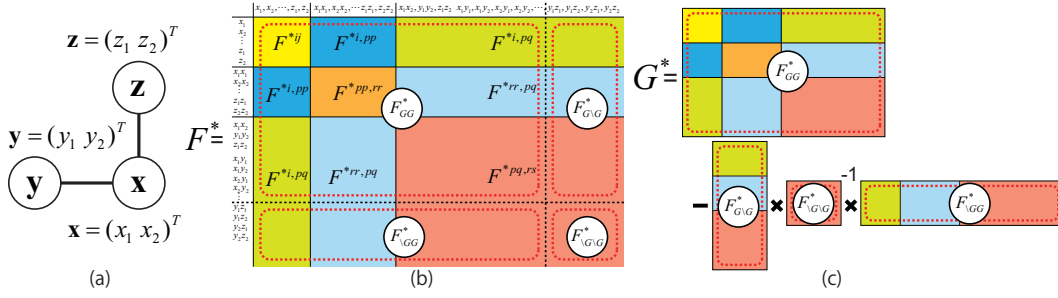

Figure 3: Here $\mathcal{V} = \{x, y, z\}$ and $\mathcal{E} = \{\{x, y\}, \{x, z\}\}$. The dimensionality of the local features is 2 ($x = (x_1, x_2)^\top$, $y = (y_1, y_2)^\top$, $z = (z_1, z_2)^\top$). A vector concatenating the local features in $\mathcal{V}$ is $v = (x_1, x_2, y_1, y_2, z_1, z_2)^\top$. Using the training data, we calculate a mean $\mu$ and a precision matrix $J$ of $v$. Using $\mu$ and $J$, the Fisher information matrix of the full Gaussian family can be calculated as in (b), whose rows and columns correspond to the elements of the expectation parameters. In (b), $F^*(\eta)$ can be partitioned into the submatrices $F_{\mathcal{G},\mathcal{G}}^*(\eta)$, $F_{\mathcal{G},\backslash\mathcal{G}}^*(\eta)$, $F_{\backslash\mathcal{G},\mathcal{G}}^*(\eta)$ and $F_{\backslash\mathcal{G},\backslash\mathcal{G}}^*(\eta)$. The Fisher information matrix of the GMRF is obtained as shown in (c) using the submatrices of $F^*(\eta)$.

matrix $J = P^{-1}$, $P = \mathbb{E}[(x - \mu)(x - \mu)^\top]$ are obtained. Since the FIM $G^*(\eta)$ of the GMRF is derived from the FIM of the full Gaussian family $F^*(\eta)$, we now calculate $F^*(\eta)$ with $\mu$ and $J$. Let $e^{*i}$, $e^{*ij}$ denote the basis vectors corresponding to $\mu_i$ and $P_{ij} + \mu_i\mu_j$ in Eq. (4) respectively. The elements of $F^*(\eta)$ are obtained by [20]: $F^{*ij}(\eta) = \langle e^{*i}, e^{*j}\rangle = J_{ij}(1 + \mu^\top J\mu) + \sum_k \mu_k J_{ki} \sum_k \mu_k J_{kj}$, $F^{*i,pq}(\eta) = \langle e^{*i}, e^{*pq}\rangle = -J_{pi}\sum_k \mu_k J_{kq} - J_{qi}\sum_k \mu_k J_{kp}$, $F^{*i,pp}(\eta) = \langle e^{*i}, e^{*pp}\rangle = -J_{pi}\sum_k \mu_k J_{kp}$, $F^{*pq,rs}(\eta) = \langle e^{*pq}, e^{*rs}\rangle = J_{ps}J_{qr} + J_{qs}J_{pr}$, $F^{*pq,rr}(\eta) = \langle e^{*pq}, e^{*rr}\rangle = J_{pr}J_{rq}$, $F^{*pp,rr}(\eta) = \langle e^{*pp}, e^{*rr}\rangle = \frac{1}{2}J_{pr}^2$.

Next we derive $G^*(\eta)$ from $F^*(\eta)$. $F^*(\eta)$ can be partitioned according to the graphs $\mathcal{G}$ and $\backslash\mathcal{G}$:

$$F^*(\eta) = \begin{pmatrix} F_{\mathcal{G},\mathcal{G}}^*(\eta) & F_{\mathcal{G},\backslash\mathcal{G}}^*(\eta) \\ F_{\backslash\mathcal{G},\mathcal{G}}^*(\eta) & F_{\backslash\mathcal{G},\backslash\mathcal{G}}^*(\eta) \end{pmatrix}. \tag{9}$$

A FIM of the GMRF $G^*(\eta)$ is obtained as the Schur complement of $F^*(\eta)$ with respect to the submatrix $F_{\backslash\mathcal{G},\backslash\mathcal{G}}^*(\eta)$ [15]:

$$G^*(\eta) = F_{\mathcal{G},\mathcal{G}}^*(\eta) - F_{\mathcal{G},\backslash\mathcal{G}}^*(\eta)\left(F_{\backslash\mathcal{G},\backslash\mathcal{G}}^*(\eta)\right)^{-1} F_{\backslash\mathcal{G},\mathcal{G}}^*(\eta). \tag{10}$$

As these calculations may be complicated, we present a simple example using a GMRF with $n = 3$ vertices, shown in Fig. 3.

However, $G^*(\eta)$ is difficult to deal with as it depends on the expectation parameters. Thus, we approximate the model space using the tangent space at the center point of all training data [20]: $G^*(\eta) \approx G^*(\eta_c)$ where $\eta_c = \sum_{i=1}^N \eta_i$, and $N$ is the number of training images. In order to embed the proper metric into the expectation parameters, we multiply $G^*(\eta_c)^{1/2}$ by $\eta$:

$$\zeta = \left(F_{\mathcal{G},\mathcal{G}}^*(\eta_c) - F_{\mathcal{G},\backslash\mathcal{G}}^*(\eta_c)\left(F_{\backslash\mathcal{G},\backslash\mathcal{G}}^*(\eta_c)\right)^{-1} F_{\backslash\mathcal{G},\mathcal{G}}^*(\eta_c)\right)^{1/2} \eta. \tag{11}$$

We call $\zeta$ Graphical Gaussian Vector (GGV). This vector is used directly to build sophisticated linear classifiers.

We have a derivation of GGV, and the algorithm for it is very simple, consisting of the following three steps: 1) calculation of local auto-correlations of local features; 2) estimation of the expectation parameters of the GMRF; and 3) embedding the distance metric (the Fisher information metric) into the expectation parameters. The calculation of GGV is given in Algorithm 1. Before the calculation of GGV, we have to estimate the FIM of GMRF by decomposing the FIM of the full Gaussian. As a consequence, we obtain one common FIM for all expectation parameters. In practice, since using all training data is infeasible to estimate the FIM, we use a subset of local features randomly sampled from training data. Note that since the calculation of the FIM is done in the preprocessing stage, it is not necessary to calculate the FIM when extracting GGVs.

**Algorithm 1** Calculation of GGV.

---
**Input:** An image region $J$, and the Fisher information matrix of the GMRF $G^*(\boldsymbol{\eta}_c)$
**Output:** GGV $\boldsymbol{\zeta}$

   1. Calculate local auto-correlations of local features:
       $\boldsymbol{\mu}_i = \frac{1}{N_J} \sum_{k \in J} \boldsymbol{x}(\boldsymbol{r}_k + \boldsymbol{a}_i), \;\; C_{i,j} = \frac{1}{N_J} \sum_{k \in J} \boldsymbol{x}(\boldsymbol{r}_k + \boldsymbol{a}_i)\boldsymbol{x}(\boldsymbol{r}_k + \boldsymbol{a}_j)^\top$
   2. Estimate the expectation parameters:
       $\boldsymbol{\eta} = (\boldsymbol{\mu}_0^\top \cdots \boldsymbol{\mu}_{n-1}^\top \; f^\top(C_{0,0}) \cdots f^\top(C_{n-1,n-1}) \; g^\top(C_{0,1}) \cdots g^\top(C_{0,n-1}))^\top$
   3. Embed the Fisher information metric into the expectation parameters:
       $\boldsymbol{\zeta} = (G^*(\boldsymbol{\eta}_c))^{1/2} \boldsymbol{\eta}$

---

# 3 Experiment

We tested our method on the standard object and scene datasets (Caltech101, Caltech256, and 15-Scenes). For the first experiment, we evaluated the effects of the graph structure (i.e. spatial information) and the FIM. As baseline methods, we used Generalized Local Correlation (GLC) [19]: $\boldsymbol{\eta}_{glc} = (\boldsymbol{\mu}_0^\top \; f^\top(C_{0,0}))^\top$ without the FIM, Local Auto-Correlation features (LAC) [21], [14]: $\boldsymbol{\eta}_{lac} = (\boldsymbol{\mu}_0^\top \; f^\top(C_{0,0}) \; g^\top(C_{0,1}) \cdots g^\top(C_{0,n-1}))^\top$ without the FIM, and the Global Gaussian with a center linear kernel (GG) [20]: $\boldsymbol{\eta}_{glc}$ with $F^*(\boldsymbol{\eta}_c)$. The comparison among these methods are shown in Table 1. Two types of graph structures were utilized for the GGVs. The first is shown in Fig. 2(c) (GGV, $n = 3$), which models a horizontal and vertical spatial layout of the local features. The second is shown in Fig. 2(d) (GGV, $n = 5$), which adds diagonal spatial layouts of the features to Fig. 2(c). We also compared L2 normalized GGVs (i.e., $\hat{\boldsymbol{\zeta}} = \boldsymbol{\zeta}/||\boldsymbol{\zeta}||$). To embed the global spatial information, we used the spatial pyramid representation with a $1 \times 1 + 2 \times 2 + 3 \times 3$ pyramid structure.

Table 1: The relationships between GLC, LAC, GG, and GGV in terms of spatial information and Fisher information metrics.

| Method | Spatial information | Fisher information metric |
|:---:|:---:|:---:|
| GLC | - | - |
| LAC | $\checkmark$ | - |
| GG | - | $\checkmark$ |
| GGV (proposed) | $\checkmark$ | $\checkmark$ |

In the second experiment, we compared GGVs with the Improved Fisher kernel (IFK) [24], [25], which is the best image representation available at the time of writing. In this experiment, we used the spatial pyramid representation with a $1 \times 1 + 2 \times 2 + 3 \times 1$ structure. The number of components $c$ in GMMs is an important parameter for IFK. We tested GMMs with $c = 32, \; 64, \; 128,$ and $256$ Gaussians to compute IFKs and compared them with GGVs.

For all datasets, SIFT features were densely sampled and were described for $16 \times 16$ patches. We downsized images if their longest side was more than 300 pixels. As the aforementioned features depend on the dimensionality of the local feature, we reduced its dimensionality using PCA and compared performance as a function of its new dimensionality. As a linear classifier, we used the multi-class Passive-Aggressive Algorithm (PA) [6].

## 3.1 Caltech101

Caltech101 is the de facto standard object-recognition dataset [10]. To evaluate the classification performance, we followed the most commonly used methodology. Fifteen images were randomly selected from all 102 categories for training purposes and the remaining images were used for testing. The classification score was averaged over 10 trials.

Before comparison between GGVs and the baselines, we evaluate the sensitivities of the sampling step of local features. The sampling step is one of the important parameters of GGV, because GGV calculates auto-correlations of the neighboring local features. In this preliminary experiment, we fix the number of vertices is 5 ($n = 5$) and the dimensionality of local feature is 32. We do not use

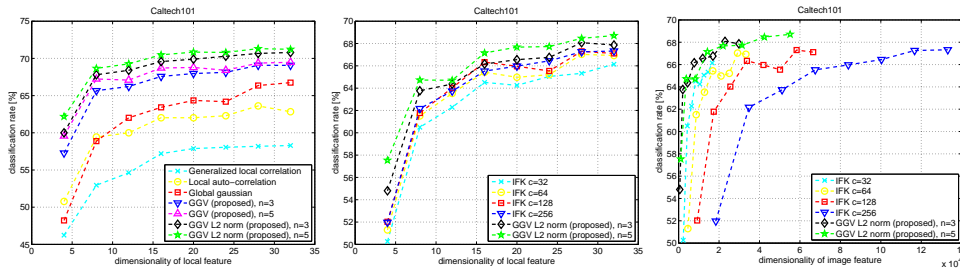

Figure 4: A comparison of classification accuracies of: (left) GGV, GLC, LAC and GG; (center) GGV and IFK with respect to the dimensionality of "local features"; (right) GGV and IFK with respect to the dimensionality of "image features" in the Caltech101 dataset.

the spatial pyramid. The results are as follows: 56.7 % (step = 4 pixels), 57.7 % (step = 6 pixels) , 57.7 % (step = 8 pixels) , 57.2 % (step = 10 pixels) , 56.5 % (step = 12 pixels). There is no clear difference between step sizes of 6 and 8 pixels. Therefore in the following experiments, we use 6 pixels sampling step for local feature extraction.

Figure 4 (left) shows the classification accuracies as a function of the dimensionality of the local features. A large dimensionality yielded better performance, and the proposed method (GGV) outperformed the other methods (GLC, LAC, and GG). By comparing GGV with LAC, and GG with GLC, it is clear that embedding the Fisher information metric improved the classification accuracy significantly. By comparing GGV with GG, as well as LAC with GLC, it can also be seen that embedding the spatial layout of local features also improved the accuracy. In a comparison between the graph structures, the four-neighbor structure (Fig. 2(d)) performed slightly better than the two-neighbor structure (Fig. 2(c)). If we compare the regular GGVs with the L2 normalized GGVs, we find that the L2 normalization improved the accuracy by almost 2 %.

In the second experiment, we compared the L2 normalized GGVs with IFKs. The results are shown in Fig. 4 (center). For all the dimensionalities and numbers of components, GGVs performed better than IFKs. Fig. 4 (right) shows the classification accuracy as a function of the dimensionality of the image features which are converted from the results shown in Fig. 4 (center). We see that GGVs achieved higher accuracy for a lower dimensionality of image features. The results were also compared against those of leading methods that use a linear classifier. The performance scores are referenced from the original papers. LLC [27] scored 65.4 % and ScSPM [28] scored 67.0 %, whereas our method achieved 71.3 % when the dimensionality of the local feature is 32 and the number of vertices is 5. Therefore, our method is better than the best available methods in this dataset, despite using a linear classifier and not requiring a codebook or descriptor matching.

## 3.2 Caltech256

Caltech256 consists of images from 256 object categories [13]. This database is significant for its large inter-class variability, as well as an intra-class variability greater than that found in Caltech101. To evaluate performance, we followed a commonly used methodology. Fifteen images were randomly selected from all categories for training purposes and the remaining images were used for testing. The classification score was averaged over 10 trials.

Figure 5 (left) shows a comparison of classification accuracies of GGV, GLC, LAC and GG. Fig. 5 (center) and (right) show comparisons of the L2 normalized GGVs and IFKs using the Caltech256 dataset with respect to the dimensionality of local features and image features, respectively. The results show the same trends as for Caltech101. Our method is better than all baseline methods and IFKs. [24] reported that IFK achieved 34.7% and [27] reported that LLC scored 34.4%, while GGV obtained 33.4%. However, a fair comparison is difficult because our method used only single-scale SIFT whereas [24] and [27] used 5-scale SIFT and 3-scale HOG, respectively. It is known that using multi-scale local features improves classification accuracies (e.g. [3]). To be fair comparison, we used 3-scale SIFT (patch size = $16 \times 16$, $24 \times 24$, $32 \times 32$) for GGV with $n = 5$, and L2 normalization. GGV with 3-scale SIFT achieved 36.2% which is better than those leading methods.

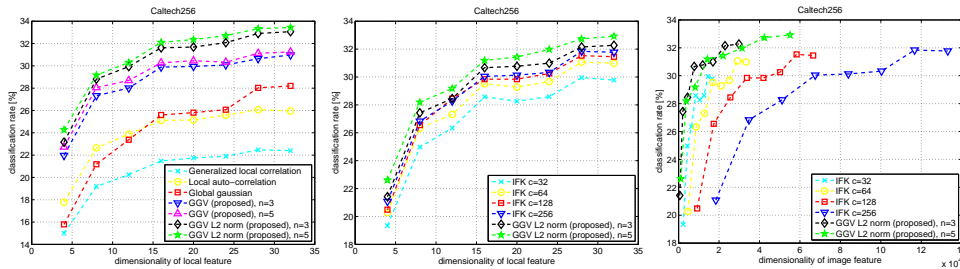

Figure 5: A comparison of classification accuracies of: (left) GGV, GLC, LAC and GG; (center) GGV and IFK with respect to the dimensionality of "local features"; (right) GGV and IFK with respect to the dimensionality of "image features" in the Caltech256 dataset.

### 3.3 15-Scenes

We experimented with 15-Scenes, a commonly used scene classification dataset [18]. We randomly selected 100 training images for each class and used the remaining samples as test data. We calculated the mean of the classification rate for each class. This score was averaged over 10 trials, where the training and test sets were randomly re-selected for each trial. This is the same methodology as that used in previous studies.

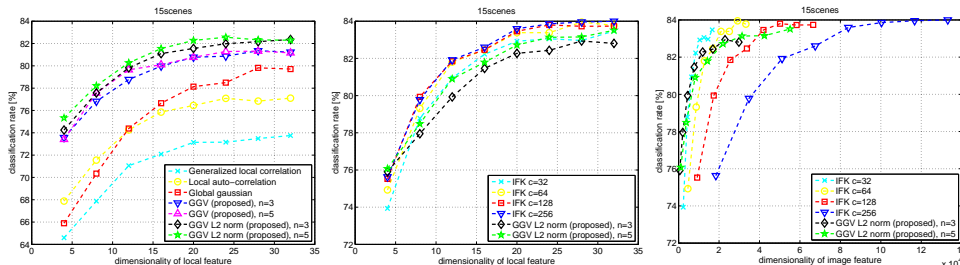

Figure 6: A comparison of classification accuracies of: (left) GGV, GLC, LAC and GG; (center) GGV and IFK with respect to the dimensionality of "local features"; (right) GGV and IFK with respect to the dimensionality of "image features" in the 15-Scenes dataset.

Figure 6 (left) shows a comparison of classification accuracies of GGV, GLC, LAC and GG using the 15-Scenes dataset. The results show similar trends as for Caltech101 and Caltech256, except that there is no difference between the scores of the graph structures. In the second experiment, the results with respect to the dimensionality of local features and image features are shown in Figs. 6 (center) and (right), respectively. In contrast to the results for Caltech101 and 256, IFKs scored slightly higher than GGVs (IFK ($c = 256, d = 32$): 84.0%, GGV ($n = 5, d = 32$ and L2 normalized): 83.5%). As the leading method, the spatial Fisher kernel [17] reported the highest score (88.1%). However, since [17] used 8-scale SIFT descriptors, which provide richer information than the single-scale SIFT descriptors we used, it is difficult to make a direct comparison.

## 4 Conclusion

In this paper, we proposed an efficient image feature called a Graphical Gaussian Vector, which uses neither codebook nor local feature matching. In the proposed method, spatial information about local features and the Fisher information metric are embedded into a feature by modeling the image as the Gaussian Markov Random Field (GMRF). Experimental results using three standard datasets demonstrated that the proposed method offers a performance that is superior or comparable to other state-of-the-art methods. The proposed image feature calculates the expectation parameters of the GMRF simply and effectively while maintaining a high classification rate.

# References

[1] S. Amari and H. Nagaoka. *Methods of Information Geometry*, volume 191 of *Translations of mathematical monographs*. American Mathematical Society, 2001.

[2] A.C. Berg, T.L. Berg, and J. Malik. Shape matching and object recognition using low distortion correspondence. In *CVPR*, 2005.

[3] L. Bo, X. Ren, and D. Fox. Kernel descriptors for visual recognition. In *NIPS*, 2010.

[4] O. Boiman, E. Shechtman, and M. Irani. In defense of nearest-neighbor based image classification. In *CVPR*, 2008.

[5] Y. Cao, C. Wang, Z. Li, L. Zhang, and L. Zhang. Spatial-bag-of-features. In *CVPR*, 2010.

[6] K. Crammer, O. Dekel, J. Keshet, S. Shalev-Shwartz, and Y. Singer. Online passive-aggressive algorithms. *JMLR*, 7:551–585, 2006.

[7] G. Csurka, C. R. Dance, L. Fan, J. Willamowski, and C. Bray. Visual categorization with bags of keypoints. In *ECCV International Workshop on SLCV*, 2004.

[8] O. Duchenne, A. Joulin, and J. Ponce. A graph-matching kernel for object categorization. In *ICCV*, 2011.

[9] J.D.R. Farquhar, S. Szedmak, H. Meng, and J. Shawe-Taylor. Improving "bag-of-keypoints" image categorisation: Generative models and pdf-kernels. Technical report, University of Southampton, 2005.

[10] L. Fei-Fei, R. Fergus, and P. Perona. Learning generative visual models from few training examples: an incremental bayesian approach tested on 101 object categories. In *CVPR, Workshop on GMBV*, 2004.

[11] R. Fergus, P. Perona, and A. Zisserman. Object class recognition by unsupervised scale-invariant learning. In *CVPR*, 2003.

[12] R. Fergus, P. Zisserman, and A. Perona. Weakly supervised scale-invariant learning of models for visual recognition. *IJCV*, 71(3):273–303, 2007.

[13] G. Griffin, A. Holub, and P. Perona. Caltech-256 object category dataset. Technical Report 7694, California Institute of Technology, 2007.

[14] T. Harada, H. Nakayama, and Y. Kuniyoshi. Improving local descriptors by embedding global and local spatial information. In *ECCV*, 2010.

[15] Jason K. Johnson. *Convex Relaxation Methods for Graphical Models: Lagrangian and Maximum Entropy Approaches*. PhD thesis, MIT, 2008.

[16] J. Kim and K. Grauman. Asymmetric region-to-image matching for comparing images with generic object categories. In *CVPR*, 2010.

[17] J. Krapac, J. Verbeek, and F. Jurie. Modeling spatial layout with fisher vectors for image categorization. In *ICCV*, 2011.

[18] S. Lazebnik, C. Schmid, and J. Ponce. Beyond bags of features: Spatial pyramid matching for recognizing natural scene categories. In *CVPR*, 2006.

[19] H. Nakayama, T. Harada, and Y. Kuniyoshi. Dense sampling low-level statistics of local features. In *CIVR* , 2009.

[20] H. Nakayama, T. Harada, and Y. Kuniyoshi. Global gaussian approach for scene categorization using information geometry. In *CVPR*, 2010.

[21] N. Otsu and T. Kurita. A new scheme for practical, flexible and intelligent vision systems. In *Proc. IAPR Workshop on Computer Vision*, 1988.

[22] F. Perronnin and C. Dance. Fisher kernels on visual vocabularies for image categorization. In *CVPR*, 2007.

[23] F. Perronnin, C. Dance, G. Csurka, and M. Bressan. Adapted vocabularies for generic visual categorization. In *ECCV*, 2006.

[24] F. Perronnin, J. Sánchez, and T. Mensink. Improving the fisher kernel for large-scale image classification. In *ECCV*, 2010.

[25] J. Sánchez and F. Perronnin. High-dimensional signature compression for large-scale image classification. In *CVPR*, 2011.

[26] C. Wallraven, B. Caputo, and A. Graf. Recognition with local features: the kernel recipe. In *ICCV*, 2003.

[27] J. Wang, J. Yang, K. Yu, F. Lv, T. Huang, and Y. Gong. Locality-constrained linear coding for image classification. In *CVPR* , 2010.

[28] J. Yang, K. Yu, Y. Gong, and T. Huang. Linear spatial pyramid matching using sparse coding for image classification. In *CVPR*, 2009.

[29] X. Zhou, K. Yu, T. Zhang, and T. S. Huang. Image classification using super-vector coding of local image descriptors. In *ECCV*, 2010.

